# Dynamic Stochastic Synapses as Computational Units

**Wolfgang Maass**
Institute for Theoretical Computer Science
Technische Universität Graz,
A–8010 Graz, Austria.
email: maass@igi.tu-graz.ac.at

**Anthony M. Zador**
The Salk Institute
La Jolla, CA 92037, USA
email: zador@salk.edu

## Abstract

In most neural network models, synapses are treated as static weights that change only on the slow time scales of learning. In fact, however, synapses are highly dynamic, and show use-dependent plasticity over a wide range of time scales. Moreover, synaptic transmission is an inherently stochastic process: a spike arriving at a presynaptic terminal triggers release of a vesicle of neurotransmitter from a release site with a probability that can be much less than one. Changes in release probability represent one of the main mechanisms by which synaptic efficacy is modulated in neural circuits. We propose and investigate a simple model for dynamic stochastic synapses that can easily be integrated into common models for neural computation. We show through computer simulations and rigorous theoretical analysis that this model for a dynamic stochastic synapse increases computational power in a nontrivial way. Our results may have implications for the processing of time-varying signals by both biological and artificial neural networks.

A synapse $S$ carries out computations on spike trains, more precisely on trains of spikes from the presynaptic neuron. Each spike from the presynaptic neuron may or may not trigger the release of a neurotransmitter-filled vesicle at the synapse. The probability of a vesicle release ranges from about 0.01 to almost 1. Furthermore this release probability is known to be strongly "history dependent" [Dobrunz and Stevens, 1997]. A spike causes an excitatory or inhibitory potential (EPSP or IPSP, respectively) in the postsynaptic neuron only when a vesicle is released.

A spike train is represented as a sequence $\underline{t}$ of firing times, i.e. as increasing sequences of numbers $t_1 < t_2 < \ldots$ from $\mathbf{R}^+ := \{z \in \mathbf{R} : z \geq 0\}$. For each spike train $\underline{t}$ the output of synapse $S$ consists of the sequence $S(\underline{t})$ of those $t_i \in \underline{t}$ on which vesicles are "released" by $S$, i.e. of those $t_i \in \underline{t}$ which cause an excitatory or inhibitory postsynaptic potential (EPSP or IPSP, respectively). The map $\underline{t} \to S(\underline{t})$ may be viewed as a stochastic function that is *computed* by synapse $S$. Alternatively one can characterize the output $S(\underline{t})$ of a synapse $S$ through its *release pattern* $\underline{q} = q_1 q_2 \ldots \in \{R, F\}^*$, where $R$ stands for release and $F$ for failure of release. For each $t_i \in \underline{t}$ one sets $q_i = R$ if $t_i \in S(\underline{t})$, and $q_i = F$ if $t_i \notin S(\underline{t})$.

## 1 Basic model

The central equation in our dynamic synapse model gives the probability $p_S(t_i)$ that the $i^{th}$ spike in a presynaptic spike train $\underline{t} = (t_1, \ldots, t_k)$ triggers the release of a vesicle at time $t_i$ at synapse $S$,

$$p_S(t_i) = 1 - e^{-C(t_i) \cdot V(t_i)} \ . \tag{1}$$

The release probability is assumed to be nonzero only for $t \in \underline{t}$, so that releases occur only when a spike invades the presynaptic terminal (*i.e.* the spontaneous release probability is assumed to be zero). The functions $C(t) \geq 0$ and $V(t) \geq 0$ describe, respectively, the states of facilitation and depletion at the synapse at time $t$ .

The dynamics of facilitation are given by

$$C(t) \ = \ C_0 + \sum_{t_i < t} c(t - t_i) \ , \tag{2}$$

where $C_0$ is some parameter $\geq 0$ that can for example be related to the resting concentration of calcium in the synapse. The exponential response function $c(s)$ models the response of $C(t)$ to a presynaptic spike that had reached the synapse at time $t - s$: $c(s) = \alpha \cdot e^{-s/\tau_C}$ , where the positive parameters $\tau_C$ and $\alpha$ give the decay constant and magnitude, respectively, of the response. The function $C$ models in an abstract way internal synaptic processes underlying presynaptic facilitation, such as the concentration of calcium in the presynaptic terminal. The particular exponential form used for $c(s)$ could arise for example if presynaptic calcium dynamics were governed by a simple first order process.

The dynamics of depletion are given by

$$V(t) \ = \ \max( \ 0 \, , \ V_0 - \sum_{t_i: \ t_i < t \text{ and } t_i \in S(\underline{t})} v(t - t_i)) \ , \tag{3}$$

for some parameter $V_0 > 0$. $V(t)$ depends on the subset of those $t_i \in \underline{t}$ with $t_i < t$ on which vesicles were actually released by the synapse, i.e. $t_i \in S(\underline{t})$. The function $v(s)$ models the response of $V(t)$ to a preceding release of the same synapse at time $t - s \leq t$ . Analogously as for $c(s)$ one may choose for $v(s)$ a function with exponential decay $v(s) = e^{-s/\tau_V}$ , where $\tau_V > 0$ is the decay constant. The function $V$ models in an abstract way internal synaptic processes that support presynaptic depression, such as depletion of the pool of readily releasable vesicles. In a more specific synapse model one could interpret $V_0$ as the maximal number of vesicles that can be stored in the readily releasable pool, and $V(t)$ as the expected number of vesicles in the readily releasable pool at time $t$.

In summary, the model of synaptic dynamics presented here is described by five parameters: $C_0$, $V_0$, $\tau_C$, $\tau_V$ and $\alpha$. The dynamics of a synaptic computation and its internal variables $C(t)$ and $V(t)$ are indicated in Fig. 1.

For low release probabilities, Eq. 1 can be expanded to first order around $r(t) := C(t) \cdot V(t) = 0$ to give

$$p_S(t_i) = C(t_i) \cdot V(t_i) + O([C(t_i) \cdot V(t_i)]^2). \tag{4}$$

Similar expressions have been widely used to describe synaptic dynamics for multiple synapses [Magleby, 1987, Markram and Tsodyks, 1996, Varela et al., 1997].

In our synapse model, we have assumed a standard exponential form for the decay of facilitation and depression (see e.g. [Magleby, 1987, Markram and Tsodyks, 1996, Varela et al., 1997, Dobrunz and Stevens, 1997]). We have further assumed a multiplicative interaction between facilitation and depletion. While this form has not been validated

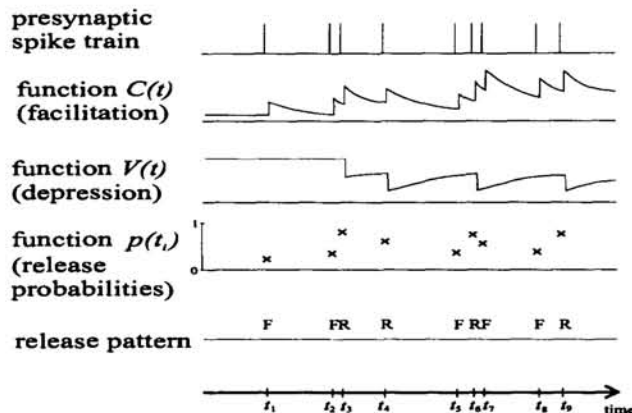

Figure 1: *Synaptic computation on a spike train $\underline{t}$, together with the temporal dynamics of the internal variables $C$ and $V$ of our model. Note that $V(t)$ changes its value only when a presynaptic spike causes release.*

at single synapses, in the limit of low release probability (see Eq. 4), it agrees with the multiplicative term employed in [Varela et al., 1997] to describe the dynamics of multiple synapses.

The assumption that release at individual release sites of a synapse is binary, *i.e.* that each release site releases 0 or 1—but not more than 1—vesicle when invaded by a spike, leads to the exponential form of Eq. 1 [Dobrunz and Stevens, 1997]. We emphasize the formal distinction between *release site* and *synapse*. A synapse might consist of several release sites in parallel, each of which has a dynamics similar to that of the stochastic "synapse model" we consider.

## 2   Results

### 2.1   Different "Weights" for the First and Second Spike in a Train

We start by investigating the range of different release probabilities $p_S(t_1), p_S(t_2)$ that a synapse $S$ can assume for the first two spikes in a given spike train. These release probabilities depend on $t_2 - t_1$ as well as on the values of the internal parameters $C_0, V_0, \tau_C, \tau_V, \alpha$ of the synapse $S$. Here we analyze the potential freedom of a synapse to choose values for $p_S(t_1)$ and $p_S(t_2)$. We show in Theorem 2.1 that the range of values for the release probabilities for the first two spikes is quite large, and that the entire attainable range can be reached through through suitable choices of $C_0$ and $V_0$.

**Theorem 2.1** *Let $\langle t_1, t_2 \rangle$ be some arbitrary spike train consisting of two spikes, and let $p_1, p_2 \in (0, 1)$ be some arbitrary given numbers with $p_2 > p_1 \cdot (1 - p_1)$. Furthermore assume that arbitrary positive values are given for the parameters $\alpha, \tau_C, \tau_V$ of a synapse $S$. Then one can always find values for the two parameters $C_0$ and $V_0$ of the synapse $S$ so that $p_S(t_1) = p_1$ and $p_S(t_2) = p_2$.*

*Furthermore the condition $p_2 > p_1 \cdot (1 - p_1)$ is necessary in a strong sense. If $p_2 \leq p_1 \cdot (1 - p_1)$ then no synapse $S$ can achieve $p_S(t_1) = p_1$ and $p_S(t_2) = p_2$ for any spike train $\langle t_1, t_2 \rangle$ and for any values of its parameters $C_0, V_0, \tau_C, \tau_V, \alpha$.*

If one associates the current sum of release probabilities of multiple synapses or release sites between two neurons $u$ and $v$ with the current value of the "connection strength" $w_{u,v}$ between two neurons in a formal neural network model, then the preceding result points

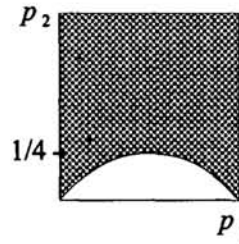

Figure 2: *The dotted area indicates the range of pairs* $\langle p_1, p_2 \rangle$ *of release probabilities for the first and second spike through which a synapse can move (for any given interspike interval) by varying its parameters* $C_0$ *and* $V_0$.

to a significant difference between the dynamics of computations in biological circuits and formal neural network models. Whereas in formal neural network models it is commonly assumed that the value of a synaptic weight stays fixed during a computation, the release probabilities of synapses in biological neural circuits may change on a fast time scale within a single computation.

## 2.2 Release Patterns for the First Three Spikes

In this section we examine the variety of release patterns that a synapse can produce for spike trains $t_1, t_2, t_3, \ldots$ with at least three spikes. We show not only that a synapse can make use of different parameter settings to produce different release patterns, but also that a synapse with a *fixed* parameter setting can respond quite differently to spike trains with different interspike intervals. Hence a synapse can serve as *pattern detector* for temporal patterns in spike trains.

It turns out that the structure of the triples of release probabilities $\langle p_S(t_1), p_S(t_2), p_S(t_3) \rangle$ that a synapse can assume is substantially more complicated than for the first two spikes considered in the previous section. Therefore we focus here on the dependence of the *most likely release pattern* $\underline{q} \in \{R, F\}^3$ on the internal synaptic parameters and on the interspike intervals $I_1 := t_2 - t_1$ and $I_2 := t_3 - t_2$. This dependence is in fact quite complex, as indicated in Fig. 3.

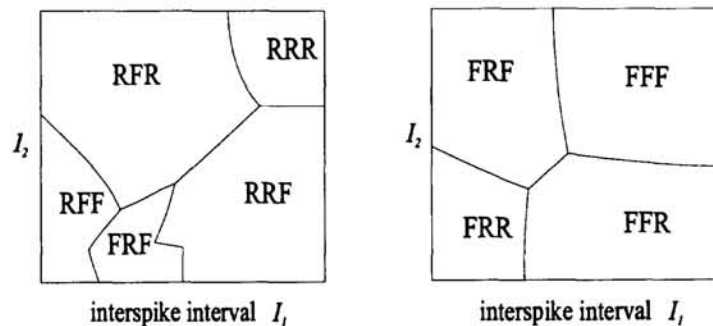

Figure 3: **(A, left)** *Most likely release pattern of a synapse in dependence of the interspike intervals* $I_1$ *and* $I_2$. *The synaptic parameters are* $C_0 = 1.5$, $V_0 = 0.5$, $\tau_C = 5$, $\tau_V = 9$, $\alpha = 0.7$. **(B, right)** *Release patterns for a synapse with other values of its parameters* $(C_0 = 0.1, V_0 = 1.8, \tau_C = 15, \tau_V = 30, \alpha = 1)$.

Fig. 3A shows the most likely release pattern for each given pair of interspike intervals $\langle I_1, I_2 \rangle$, given a particular *fixed* set of synaptic parameters. One can see that a synapse with fixed parameter values is likely to respond quite differently to spike trains with different interspike intervals. For example even if one just considers spike trains with $I_1 = I_2$ one moves in Fig. 3A through 3 different release patterns that take their turn in becoming the most likely release pattern when $I_1$ varies. Similarly, if one only considers spike trains with a fixed time interval $t_3 - t_1 = I_1 + I_2 = \Delta$, but with different positions of the second spike within this time interval of length $\Delta$, one sees that the most likely release pattern is quite sensitive to the position of the second spike within this time interval $\Delta$. Fig. 3B shows that a different set of synaptic parameters gives rise to a completely different assignment of release patterns.

We show in the next Theorem that the boundaries between the zones in these figures are "plastic": by changing the values of $C_0, V_0, \alpha$ the synapse can move the zone for most of the release patterns $\underline{q}$ to any given point $\langle I_1, I_2 \rangle$. This result provides another example for a new type of synaptic plasticity that can no longer be described in terms of a decrease or increase of the synaptic "weight".

**Theorem 2.2** *Assume that an arbitrary number $p \in (0, 1)$ and an arbitrary pattern $\langle I_1, I_2 \rangle$ of interspike intervals is given. Furthermore assume that arbitrary fixed positive values are given for the parameters $\tau_C$ and $\tau_V$ of a synapse $S$. Then for any pattern $\underline{q} \in \{R, F\}^3$ except RRF, FFR one can assign values to the other parameters $\alpha, C_0, V_0$ of this synapse $S$ so that the probability of release pattern $\underline{q}$ for a spike train with interspike intervals $I_1, I_2$ becomes larger than $p$.*

It is shown in the full version of this paper [Maass and Zador, 1997] that it is not possible to make the release patterns RRF and FFR arbitrarily likely for any given spike train with interspike intervals $\langle I_1, I_2 \rangle$ .

## 2.3   Computing with Firing Rates

So far we have considered the effect of short trains of two or three presynaptic spikes on synaptic release probability. Our next result (cf. Fig.5) shows that also two longer Poisson spike trains that represent the *same* firing rate can produce quite different numers of synaptic releases, depending on the synaptic parameters. To emphasize that this is due to the pattern of interspike intervals, and not simply to the number of spikes, we compared the outputs in response to two Poisson spike trains $A$ and $B$ with the same number (10) of spikes. These examples indicate that even in the context of rate coding, synaptic efficacy may not be well described in terms of a single scalar parameter $w$.

## 2.4   Burst Detection

Here we show that the computational power of a spiking (e.g. integrate-and-fire) neuron with stochastic dynamic synapses is strictly larger than that of a spiking neuron with traditional "static" synapses (cf Lisman, 1997). Let $T$ be a some given time window, and consider the computational task of detecting whether at least one of $n$ presynaptic neurons $a_1, \ldots, a_n$ fire at least twice during $T$ ("burst detection"). To make this task computationally feasible we assume that none of the neurons $a_1, \ldots, a_n$ fires outside of this time window.

**Theorem 2.3** *A spiking neuron $v$ with dynamic stochastic synapses can solve this burst detection task (with arbitrarily high reliability). On the other hand no spiking neuron with static synapses can solve this task (for any assignment of "weights" to its synapses).* [1]

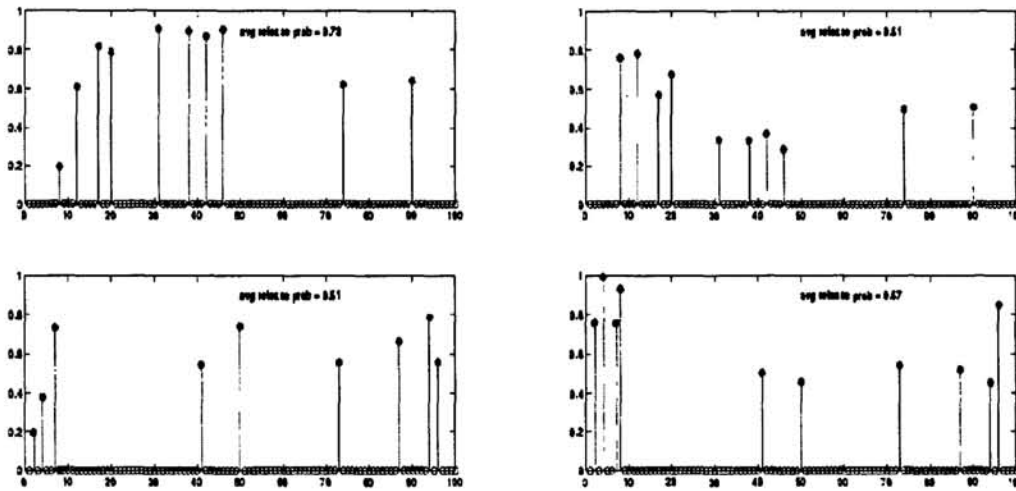

Figure 4: *Release probabilities of two synapses for two Poisson spike trains A and B with 10 spikes each. The release probabilities for the first synapse are shown on the left hand side, and for the second synapse on the right hand side. For both synapses the release probabilities for spike train A are shown at the top, and for spike train B at the bottom. The first synapse has for spike train A a 22 % higher average release probability, whereas the second synapse has for spike train B a 16 % higher average release probability. Note that the fourth spike in spike train B has for the first synapse a release probability of nearly zero and so is not visible.*

## 2.5 Translating Interval Coding into Population Coding

Assume that information is encoded in the length $I$ of the interspike interval between the times $t_1$ and $t_2$ when a certain neuron $v$ fires, and that different motor responses need to be initiated depending on whether $I < a$ or $I > a$, where $a$ is some given parameter (c.f. [Hopfield, 1995]). For that purpose it would be useful to translate the information encoded in the interspike interval $I$ into the firing activity of populations of neurons ("population coding"). Fig. 5 illustrates a simple mechanism for that task based on dynamic synapses. The synaptic parameters are chosen so that facilitation dominates (i.e., $C_0$ should be small and $\alpha$ large) at synapses between neuron $v$ and the postsynaptic population of neurons. The release probability for the first spike is then close to 0, whereas the release probability for the second spike is fairly large if $I < a$ and significantly smaller if $I$ is substantially larger than $a$. If the resulting firing activity of the postsynaptic neurons is positively correlated with the total number of releases of these synapses, then their population response is also positively correlated with the length of the interspike interval $I$.

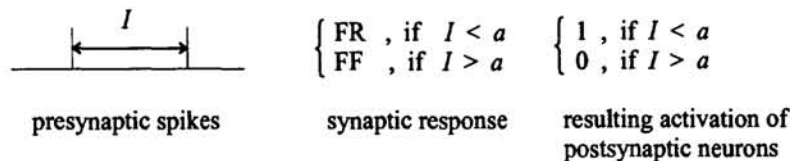

$$\left\{ \begin{array}{ll} FR & , \text{ if } I < a \\ FF & , \text{ if } I > a \end{array} \right. \qquad \left\{ \begin{array}{ll} 1 & , \text{ if } I < a \\ 0 & , \text{ if } I > a \end{array} \right.$$

presynaptic spikes      synaptic response      resulting activation of postsynaptic neurons

Figure 5: *A mechanism for translating temporal coding into population coding.*

# 3 Discussion

We have explored computational implications of a dynamic stochastic synapse model. Our model incorporates several features of biological synapses usually omitted in the connections or weights conventionally used in artificial neural network models. Our main result is that a neural circuit in which connections are dynamic has fundamentally greater power than one in which connections are static. We refer to [Maass and Zador, 1997] for details. Our results may have implications for computation in both biological and artificial neural networks, and particularly for the processing of signals with interesting temporal structure.

Several groups have recently proposed a computational role for one form of use-dependent short term synaptic plasticity [Abbott et al., 1997, Tsodyks and Markram, 1997]. They showed that, under the experimental conditions tested, synaptic depression (of a form analogous to $V(t)$ in our Eq. (3) can implement a form of gain control in which the steady-state synaptic output is independent of the input firing rate over a wide range of firing rates. We have adopted a more general approach in which, rather than focussing on a particular role for short term plasticity, we allow the dynamic synapse parameters to vary. This approach is analogous to that adopted in the study of artificial neural networks, in which few if any constraints are placed on the connections between units. In our more general framework, standard neural network tasks such as supervised and unsupervised learning can be formulated (*see also* [Liaw and Berger, 1996]). Indeed, a backpropagation-like gradient descent algorithm can be used to adjust the parameters of a network connected by dynamic synapses (Zador and Maass, *in preparation*). The advantages of dynamic synapses may become most apparent in the processing of time-varying signals.

## Footnotes

[1] We assume here that neuronal transmission delays differ by less than $(n - 1) \cdot T$, where by *transmission delay* we refer to the temporal delay between the firing of the presynaptic neuron and its effect on the postsynaptic target.

# References

[Abbott et al., 1997] Abbott, L., Varela, J., Sen, K., and S.B., N. (1997). Synaptic depression and cortical gain control. *Science*, 275:220–4.

[Dobrunz and Stevens, 1997] Dobrunz, L. and Stevens, C. (1997). Heterogeneity of release probability, facilitation and depletion at central synapses. *Neuron*, 18:995–1008.

[Hopfield, 1995] Hopfield, J. (1995). Pattern recognition computation using action potential timing for stimulus representation. *Nature*, 376:33–36.

[Liaw and Berger, 1996] Liaw, J.-S. and Berger, T. (1996). Dynamic synapse: A new concept of neural representation and computation. *Hippocampus*, 6:591–600.

[Lisman, 1997] Lisman, J. (1997). Bursts as a unit of neural information: making unreliable synapses reliable. *TINS*, 20:38–43.

[Maass and Zador, 1997] Maass, W. and Zador, A. (1997). Dynamic stochastic synapses as computational units. *http://www.sloan.salk.edu/~zador/publications.html* .

[Magleby, 1987] Magleby, K. (1987). Short term synaptic plasticity. In Edelman, G. M., Gall, W. E., and Cowan, W. M., editors, *Synaptic function*. Wiley, New York.

[Markram and Tsodyks, 1996] Markram, H. and Tsodyks, M. (1996). Redistribution of synaptic efficacy between neocortical pyramidal neurons. *Nature*, 382:807–10.

[Stevens and Wang, 1995] Stevens, C. and Wang, Y. (1995). Facilitation and depression at single central synapses. *Neuron*, 14:795–802.

[Tsodyks and Markram, 1997] Tsodyks, M. and Markram, H. (1997). The neural code between neocortical pyramidal neurons depends on neurotransmitter release probability. *Proc. Natl. Acad. Sci.*, 94:719–23.

[Varela et al., 1997] Varela, J. A., Sen, K., Gibson, J., Fost, J., Abbott, L. F., and Nelson, S. B. (1997). A quantitative description of short-term plasticity at excitatory synapses in layer 2/3 of rat primary visual cortex. *J. Neurosci*, 17:7926–7940.
